# Sensory Adaptation within a Bayesian Framework for Perception

**Alan A. Stocker**[*] and **Eero P. Simoncelli**

Howard Hughes Medical Institute and
Center for Neural Science
New York University

## Abstract

We extend a previously developed Bayesian framework for perception to account for sensory adaptation. We first note that the perceptual effects of adaptation seems inconsistent with an adjustment of the internally represented prior distribution. Instead, we postulate that adaptation increases the signal-to-noise ratio of the measurements by adapting the operational range of the measurement stage to the input range. We show that this changes the likelihood function in such a way that the Bayesian estimator model can account for reported perceptual behavior. In particular, we compare the model's predictions to human motion discrimination data and demonstrate that the model accounts for the commonly observed perceptual adaptation effects of repulsion and enhanced discriminability.

## 1  Motivation

A growing number of studies support the notion that humans are nearly optimal when performing perceptual estimation tasks that require the combination of sensory observations with *a priori* knowledge. The Bayesian formulation of these problems defines the optimal strategy, and provides a principled yet simple computational framework for perception that can account for a large number of known perceptual effects and illusions, as demonstrated in sensorimotor learning [1], cue combination [2], or visual motion perception [3], just to name a few of the many examples.

Adaptation is a fundamental phenomenon in sensory perception that seems to occur at all processing levels and modalities. A variety of computational principles have been suggested as explanations for adaptation. Many of these are based on the concept of maximizing the sensory information an observer can obtain about a stimulus despite limited sensory resources [4, 5, 6]. More mechanistically, adaptation can be interpreted as the attempt of the sensory system to adjusts its (limited) dynamic range such that it is maximally informative with respect to the statistics of the stimulus. A typical example is observed in the retina, which manages to encode light intensities that vary over nine orders of magnitude using ganglion cells whose dynamic range covers only two orders of magnitude. This is achieved by adapting to the local mean as well as higher order statistics of the visual input over short time-scales [7].

---

[*]corresponding author.

If a Bayesian framework is to provide a valid computational explanation of perceptual processes, then it needs to account for the behavior of a perceptual system, regardless of its adaptation state. In general, adaptation in a sensory estimation task seems to have two fundamental effects on subsequent perception:

- *Repulsion:* The estimate of parameters of subsequent stimuli are repelled by those of the adaptor stimulus, *i.e.* the perceived values for the stimulus variable that is subject to the estimation task are more distant from the adaptor value after adaptation. This repulsive effect has been reported for perception of visual speed (*e.g.* [8, 9]), direction-of-motion [10], and orientation [11].

- *Increased sensitivity:* Adaptation increases the observer's discrimination ability around the adaptor (*e.g.* for visual speed [12, 13]), however it also seems to decrease it further away from the adaptor as shown in the case of direction-of-motion discrimination [14].

In this paper, we show that these two perceptual effects can be explained within a Bayesian estimation framework of perception. Note that our description is at an abstract functional level - we do not attempt to provide a computational model for the underlying *mechanisms* responsible for adaptation, and this clearly separates this paper from other work which might seem at first glance similar [e.g., 15].

## 2   Adaptive Bayesian estimator framework

Suppose that an observer wants to estimate a property of a stimulus denoted by the variable $\theta$, based on a measurement $m$. In general, the measurement can be vector-valued, and is corrupted by both internal and external noise. Hence, combining the noisy information gained by the measurement $m$ with *a priori* knowledge about $\theta$ is advantageous. According to Bayes' rule

$$p(\theta|m) = \frac{1}{\alpha}p(m|\theta)p(\theta) \ . \tag{1}$$

That is, the probability of stimulus value $\theta$ given $m$ (*posterior*) is the product of the *likelihood* $p(m|\theta)$ of the particular measurement and the *prior* $p(\theta)$. The normalization constant $\alpha$ serves to ensure that the posterior is a proper probability distribution. Under the assumption of a squared-error loss function, the optimal estimate $\hat{\theta}(m)$ is the mean of the posterior, thus

$$\hat{\theta}(m) = \int_0^\infty \theta \ p(\theta|m) \ d\theta \ . \tag{2}$$

Note that $\hat{\theta}(m)$ describes an estimate for a single measurement $m$. As discussed in [16], the measurement will vary stochastically over the course of many exposures to the same stimulus, and thus the estimator will also vary. We return to this issue in Section 3.2.

Figure 1a illustrates a Bayesian estimator, in which the shape of the (arbitrary) prior distribution leads on average to a shift of the estimate toward a lower value of $\theta$ than the true stimulus value $\theta_{\text{stim}}$. The likelihood and the prior are the fundamental constituents of the Bayesian estimator model. Our goal is to describe how adaptation alters these constituents so as to account for the perceptual effects of repulsion and increased sensitivity.

### Adaptation does not change the prior ...

An intuitively sensible hypothesis is that adaptation changes the prior distribution. Since the prior is meant to reflect the knowledge the observer has about the distribution of occurrences of the variable $\theta$ in the world, repeated viewing of stimuli with the same parameter

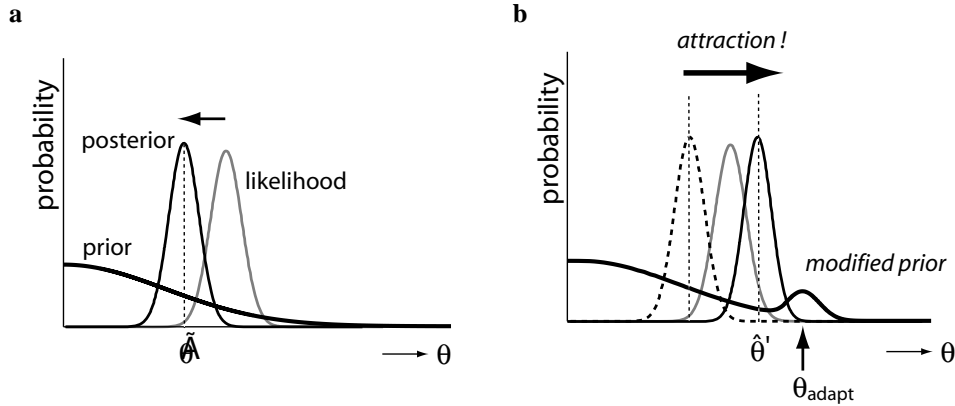

Figure 1: *Hypothetical model in which adaptation alters the prior distribution.* a) Un-adapted Bayesian estimation configuration in which the prior leads to a shift of the estimate $\hat{\theta}$, relative to the stimulus parameter $\theta_{\mathrm{stim}}$. Both the likelihood function and the prior distribution contribute to the exact value of the estimate $\hat{\theta}$ (mean of the posterior). b) Adaptation acts by increasing the prior distribution around the value, $\theta_{\mathrm{adapt}}$, of the adapting stimulus parameter. Consequently, an subsequent estimate $\hat{\theta}'$ of the same stimulus parameter value $\theta_{\mathrm{stim}}$ is *attracted* toward the adaptor. This is the opposite of observed perceptual effects, and we thus conclude that adjustments of the prior in a Bayesian model do not account for adaptation.

value $\theta_{\mathrm{adapt}}$ should presumably increase the prior probability in the vicinity of $\theta_{\mathrm{adapt}}$. Figure 1b schematically illustrates the effect of such a change in the prior distribution. The estimated (perceived) value of the parameter under the adapted condition is *attracted* to the adapting parameter value. In order to account for observed perceptual repulsion effects, the prior would have to *decrease* at the location of the adapting parameter, a behavior that seems fundamentally inconsistent with the notion of a prior distribution.

### ... but increases the reliability of the measurements

Since a change in the prior distribution is not consistent with repulsion, we are led to the conclusion that adaptation must change the likelihood function. But why, and how should this occur?

In order to answer this question, we reconsider the functional purpose of adaptation. We assume that adaptation acts to allocate more resources to the representation of the parameter values in the vicinity of the adaptor [4], resulting in a local increase in the signal-to-noise ratio (SNR). This can be accomplished, for example, by dynamically adjusting the operational range to the statistics of the input. This kind of increased operational *gain* around the adaptor has been effectively demonstrated in the process of retinal adaptation [17]. In the context of our Bayesian estimator framework, and restricting to the simple case of a scalar-valued measurement, adaptation results in a narrower conditional probability density $p(m|\theta)$ in the immediate vicinity of the adaptor, thus an increase in the reliability of the measurement $m$. This is offset by a broadening of the conditional probability density $p(m|\theta)$ in the region beyond the adaptor vicinity (we assume that total resources are conserved, and thus an increase around the adaptor must necessarily lead to a decrease elsewhere).

Figure 2 illustrates the effect of this local increase in signal-to-noise ratio on the likeli-

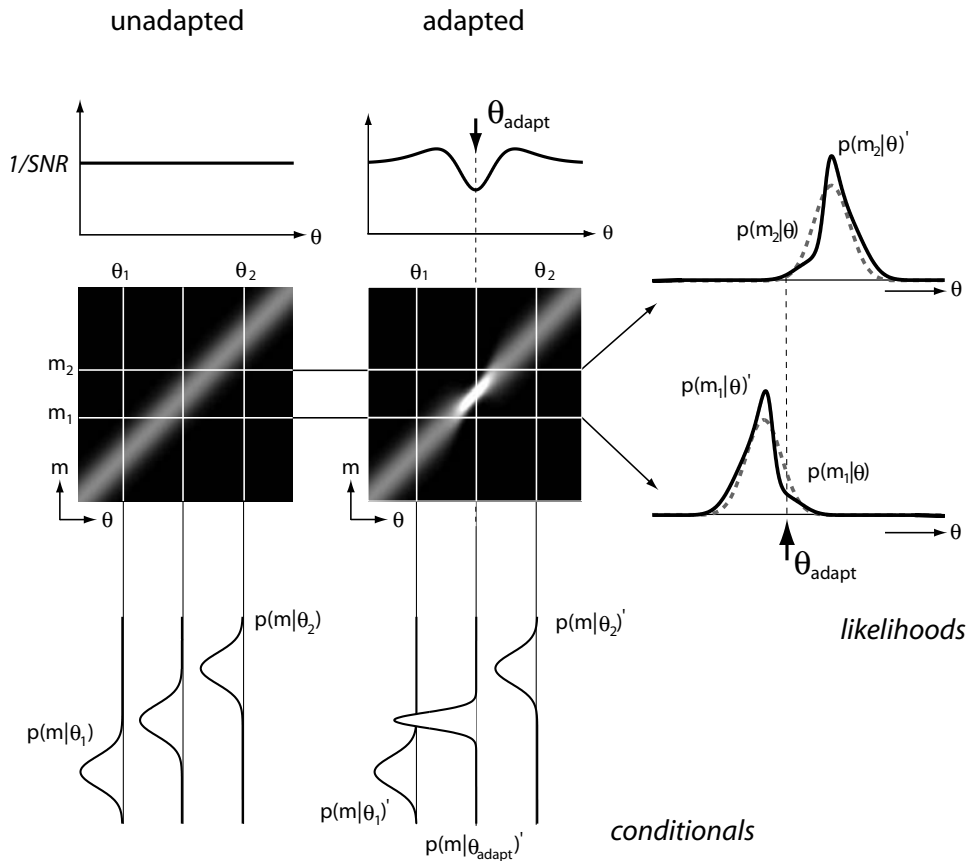

Figure 2: *Measurement noise, conditionals and likelihoods.* The two-dimensional conditional density, $p(m|\theta)$, is shown as a grayscale image for both the unadapted and adapted cases. We assume here that adaptation increases the reliability (SNR) of the measurement around the parameter value of the adaptor. This is balanced by a decrease in SNR of the measurement further away from the adaptor. Because the likelihood is a function of $\theta$ (horizontal slices, shown plotted at right), this results in an *asymmetric* change in the likelihood that is in agreement with a repulsive effect on the estimate.

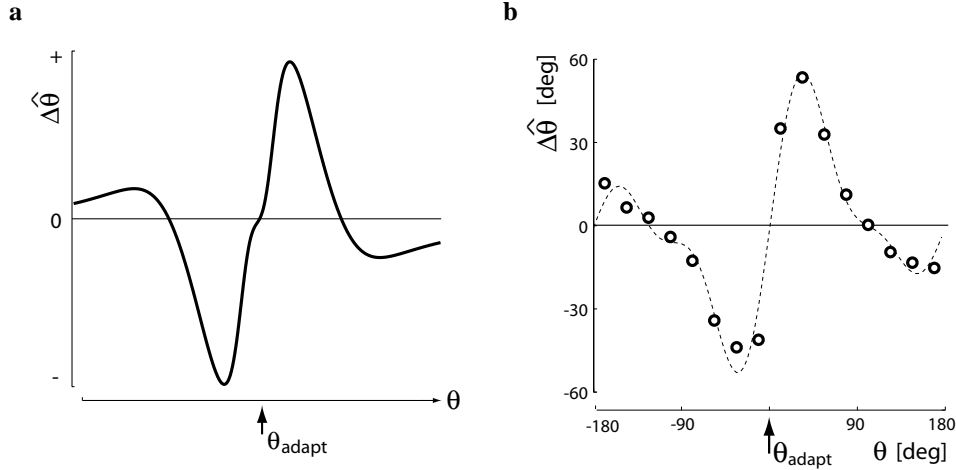

Figure 3: *Repulsion: Model predictions vs. human psychophysics.* a) Difference in perceived direction in the pre- and post-adaptation condition, as predicted by the model. Postadaptive percepts of motion direction are repelled away from the direction of the adaptor. b) Typical human subject data show a qualitatively similar repulsive effect. Data (and fit) are replotted from [10].

hood function. The two gray-scale images represent the conditional probability densities, $p(m|\theta)$, in the unadapted and the adapted state. They are formed by assuming additive noise on the measurement $m$ of constant variance (unadapted) or with a variance that decreases symmetrically in the vicinity of the adaptor parameter value $\theta_{\text{adapt}}$, and grows slightly in the region beyond. In the unadapted state, the likelihood is convolutional and the shape and variance are equivalent to the distribution of measurement noise. However, in the adapted state, because the likelihood is a function of $\theta$ (horizontal slice through the conditional surface) it is no longer convolutional around the adaptor. As a result, the mean is pushed away from the adaptor, as illustrated in the two graphs on the right. Assuming that the prior distribution is fairly smooth, this repulsion effect is transferred to the posterior distribution, and thus to the estimate.

## 3 Simulation Results

We have qualitatively demonstrated that an increase in the measurement reliability around the adaptor is consistent with the repulsive effects commonly seen as a result of perceptual adaptation. In this section, we simulate an adapted Bayesian observer by assuming a simple model for the changes in signal-to-noise ratio due to adaptation. We address both repulsion and changes in discrimination threshold. In particular, we compare our model predictions with previously published data from psychophysical experiments examining human perception of motion direction.

### 3.1 Repulsion

In the unadapted state, we assume the measurement noise to be additive and normally distributed, and constant over the whole measurement space. Thus, assuming that $m$ and $\theta$ live in the same space, the likelihood is a Gaussian of constant width. In the adapted state, we assume a simple functional description for the variance of the measurement noise around the adapter. Specifically, we use a constant plus a difference of two Gaussians,

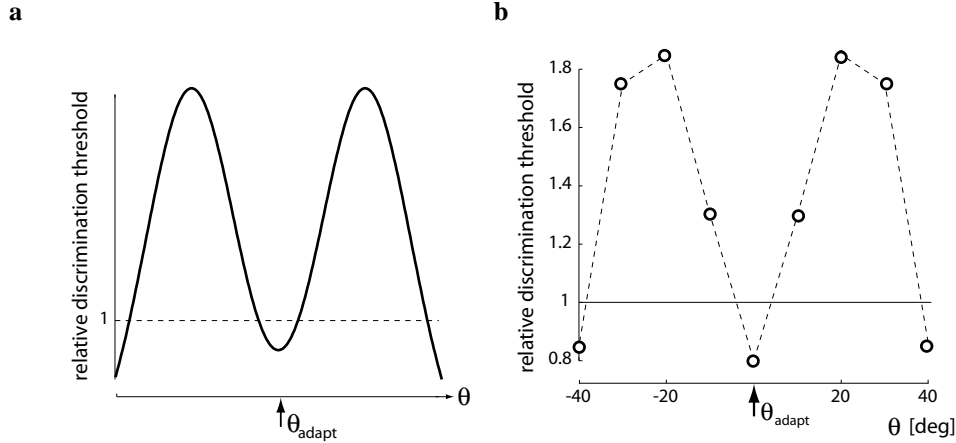

Figure 4: *Discrimination thresholds: Model predictions vs. human psychophysics.* a) The model predicts that thresholds for direction discrimination are reduced at the adaptor. It also predicts two side-lobes of increased threshold at further distance from the adaptor. b) Data of human psychophysics are in qualitative agreement with the model. Data are replotted from [14] (see also [11]).

each having equal area, with one twice as broad as the other (see Fig. 2). Finally, for simplicity, we assume a flat prior, but any reasonable smooth prior would lead to results that are qualitatively similar. Then, according to (2) we compute the predicted estimate of motion direction in both the unadapted and the adapted case.

Figure 3a shows the predicted difference between the pre- and post-adaptive average estimate of direction, as a function of the stimulus direction, $\theta_{\text{stim}}$. The adaptor is indicated with an arrow. The repulsive effect is clearly visible. For comparison, Figure 3b shows human subject data replotted from [10]. The perceived motion direction of a grating was estimated, under both adapted and unadapted conditions, using a two-alternative-forced-choice experimental paradigm. The plot shows the change in perceived direction as a function of test stimulus direction relative to that of the adaptor. Comparison of the two panels of Figure 3 indicate that despite the highly simplified construction of the model, the prediction is quite good, and even includes the small but consistent repulsive effects observed 180 degrees from the adaptor.

### 3.2 Changes in discrimination threshold

Adaptation also changes the ability of human observers to discriminate between the direction of two different moving stimuli. In order to model discrimination thresholds, we need to consider a Bayesian framework that can account not only for the mean of the estimate but also its variability. We have recently developed such a framework, and used it to quantitatively constrain the likelihood and the prior from psychophysical data [16]. This framework accounts for the effect of the measurement noise on the variability of the estimate $\hat{\theta}$. Specifically, it provides a characterization of the *distribution* $p(\hat{\theta}|\theta_{\text{stim}})$ of the estimate for a given stimulus direction in terms of its expected value and its variance as a function of the measurement noise. As in [16] we write

$$\text{var}\langle\hat{\theta}|\theta_{\text{stim}}\rangle \;\;=\;\; \text{var}\langle m\rangle(\frac{\partial\hat{\theta}(m)}{\partial m})^2|_{m=\theta_{\text{stim}}} \;. \tag{3}$$

Assuming that discrimination threshold is proportional to the standard deviation,

$\sqrt{\text{var}\langle\hat{\theta}|\theta_{\text{stim}}\rangle}$, we can now predict how discrimination thresholds should change after adaptation. Figure 4a shows the predicted change in discrimination thresholds relative to the unadapted condition for the same model parameters as in the repulsion example (Figure 3a). Thresholds are slightly reduced at the adaptor, but increase symmetrically for directions further away from the adaptor. For comparison, Figure 4b shows the relative change in discrimination thresholds for a typical human subject [14]. Again, the behavior of the human observer is qualitatively well predicted.

## 4   Discussion

We have shown that adaptation can be incorporated into a Bayesian estimation framework for human sensory perception. Adaptation seems unlikely to manifest itself as a change in the internal representation of prior distributions, as this would lead to perceptual bias effects that are opposite to those observed in human subjects. Instead, we argue that adaptation leads to an increase in reliability of the measurement in the vicinity of the adapting stimulus parameter. We show that this change in the measurement reliability results in changes of the likelihood function, and that an estimator that utilizes this likelihood function will exhibit the commonly-observed adaptation effects of repulsion and changes in discrimination threshold. We further confirm our model by making quantitative predictions and comparing them with known psychophysical data in the case of human perception of motion direction.

Many open questions remain. The results demonstrated here indicate that a resource allocation explanation is consistent with the functional effects of adaptation, but it seems unlikely that theory alone can lead to a unique quantitative prediction of the detailed form of these effects. Specifically, the constraints imposed by biological implementation are likely to play a role in determining the changes in measurement noise as a function of adaptor parameter value, and it will be important to characterize and interpret neural response changes in the context of our framework. Also, although we have argued that changes in the prior seem inconsistent with adaptation effects, it may be that such changes do occur but are offset by the likelihood effect, or occur only on much longer timescales.

Last, if one considers sensory perception as the result of a cascade of successive processing stages (with both feedforward and feedback connections), it becomes necessary to expand the Bayesian description to describe this cascade [e.g., 18, 19]. For example, it may be possible to interpret this cascade as a sequence of Bayesian estimators, in which the measurement of each stage consists of the estimate computed at the previous stage. Adaptation could potentially occur in each of these processing stages, and it is of fundamental interest to understand how such a cascade can perform useful stable computations despite the fact that each of its elements is constantly readjusting its response properties.

## References

[1] K. Körding and D. Wolpert. Bayesian integration in sensorimotor learning. *Nature*, 427(15):244–247, January 2004.

[2] D C Knill and W Richards, editors. *Perception as Bayesian Inference*. Cambridge University Press, 1996.

[3] Y. Weiss, E. Simoncelli, and E. Adelson. Motion illusions as optimal percept. *Nature Neuroscience*, 5(6):598–604, June 2002.

[4] H.B. Barlow. *Vision: Coding and Efficiency*, chapter A theory about the functional role and synaptic mechanism of visual after-effects, pages 363–375. Cambridge University Press., 1990.

[5] M.J. Wainwright. Visual adaptation as optimal information transmission. *Vision Research*, 39:3960–3974, 1999.

[6] N. Brenner, W. Bialek, and R. de Ruyter van Steveninck. Adaptive rescaling maximizes information transmission. *Neuron*, 26:695–702, June 2000.

[7] S.M. Smirnakis, M.J. Berry, D.K. Warland, W. Bialek, and M. Meister. Adaptation of retinal processing to image contrast and spatial scale. *Nature*, 386:69–73, March 1997.

[8] P. Thompson. Velocity after-effects: the effects of adaptation to moving stimuli on the perception of subsequently seen moving stimuli. *Vision Research*, 21:337–345, 1980.

[9] A.T. Smith. Velocity coding: evidence from perceived velocity shifts. *Vision Research*, 25(12):1969–1976, 1985.

[10] P. Schrater and E. Simoncelli. Local velocity representation: evidence from motion adaptation. *Vision Research*, 38:3899–3912, 1998.

[11] C.W. Clifford. Perceptual adaptation: motion parallels orientation. *Trends in Cognitive Sciences*, 6(3):136–143, March 2002.

[12] C. Clifford and P. Wenderoth. Adaptation to temporal modulaton can enhance differential speed sensitivity. *Vision Research*, 39:4324–4332, 1999.

[13] A. Kristjansson. Increased sensitivity to speed changes during adaptation to first-order, but not to second-order motion. *Vision Research*, 41:1825–1832, 2001.

[14] R.E. Phinney, C. Bowd, and R. Patterson. Direction-selective coding of stereoscopic (cyclopean) motion. *Vision Research*, 37(7):865–869, 1997.

[15] N.M. Grzywacz and R.M. Balboa. A Bayesian framework for sensory adaptation. *Neural Computation*, 14:543–559, 2002.

[16] A.A. Stocker and E.P. Simoncelli. Constraining a Bayesian model of human visual speed perception. In Lawrence K. Saul, Yair Weiss, and Léon Bottou, editors, *Advances in Neural Information Processing Systems NIPS 17*, pages 1361–1368, Cambridge, MA, 2005. MIT Press.

[17] D. Tranchina, J. Gordon, and R.M. Shapley. Retinal light adaptation – evidence for a feedback mechanism. *Nature*, 310:314–316, July 1984.

[18] S. Deneve. Bayesian inference in spiking neurons. In Lawrence K. Saul, Yair Weiss, and Léon Bottou, editors, *Adv. Neural Information Processing Systems (NIPS*04)*, vol 17, Cambridge, MA, 2005. MIT Press.

[19] R. Rao. Hierarchical Bayesian inference in networks of spiking neurons. In Lawrence K. Saul, Yair Weiss, and Léon Bottou, editors, *Adv. Neural Information Processing Systems (NIPS*04)*, vol 17, Cambridge, MA, 2005. MIT Press.
